# Probabilistic amplitude and frequency demodulation

**Richard E. Turner**[*]
Computational and Biological Learning Lab, Department of Engineering
University of Cambridge, Trumpington Street, Cambridge, CB2 1PZ, UK
ret26@cam.ac.uk

**Maneesh Sahani**
Gatsby Computational Neuroscience Unit, University College London
Alexandra House, 17 Queen Square, London, WC1N 3AR, UK
maneesh@gatsby.ucl.ac.uk

## Abstract

A number of recent scientific and engineering problems require signals to be decomposed into a product of a slowly varying positive envelope and a quickly varying carrier whose instantaneous frequency also varies slowly over time. Although signal processing provides algorithms for so-called amplitude- and frequency-demodulation (AFD), there are well known problems with all of the existing methods. Motivated by the fact that AFD is ill-posed, we approach the problem using probabilistic inference. The new approach, called probabilistic amplitude and frequency demodulation (PAFD), models instantaneous frequency using an auto-regressive generalization of the von Mises distribution, and the envelopes using Gaussian auto-regressive dynamics with a positivity constraint. A novel form of expectation propagation is used for inference. We demonstrate that although PAFD is computationally demanding, it outperforms previous approaches on synthetic and real signals in clean, noisy and missing data settings.

## 1 Introduction

Amplitude and frequency demodulation (AFD) is the process by which a signal ($y_t$) is decomposed into the product of a slowly varying envelope or amplitude component ($a_t$) and a quickly varying sinusoidal carrier ($\cos(\phi_t)$), that is $y_t = a_t \cos(\phi_t)$. In its general form this is an ill-posed problem [1], and so algorithms must impose implicit or explicit assumptions about the form of carrier and envelope to realise a solution. In this paper we make the standard assumption that the amplitude variables are slowly varying positive variables, and the derivatives of the carrier phase, $\omega_t = \phi_t - \phi_{t-1}$ called the instantaneous frequencies (IFs), are also slowly varying variables.

It has been argued that the subbands of speech are well characterised by such a representation [2, 3] and so AFD has found a range of applications in audio processing including audio coding [4, 2], speech enhancement [5] and source separation [6], and it is used in hearing devices [5]. AFD has been used as a scientific tool to investigate the perception of sounds [7]. AFD is also of importance in neural signal processing applications. Aggregate field measurements such as those collected at the scalp by electroencephalography (EEG) or within tissue as local field potentials often exhibit transient sharp spectral lines at characteristic frequencies. Within each such band, both the amplitude of the oscillation and the precise center frequencies may vary with time; and both of these phenomena may reveal important elements of the mechanism by which the field oscillation arises.

---

[*]Richard Turner would like to thank the Laboratory for Computational Vision, New York University, New York, NY 10003-6603, USA, where he carried out this research.

Despite the fact that AFD has found a wide range of important applications, there are well-known problems with existing AFD algorithms [8, 1, 9, 10, 5]. Because of these problems, the Hilbert method, which recovers an amplitude from the magnitude of the analytic signal, is still considered to be the benchmark despite a number of limitations [11, 12]. In this paper, we show examples of demodulation of synthetic, audio, and hippocampal theta rhythm signals using various AFD techniques that highlights some of the anomalies associated with existing methods.

Motivated by the deficiencies in the existing methods this paper develops a probabilistic form of AFD. This development begins in the next section where we reinterpret two existing probabilistic algorithms in the context of AFD. The limitations of these methods suggest an improved model (section 2) which we demonstrate on a range of synthetic and natural signals (sections 4 and 5).

## 1.1 Simple models for probabilistic amplitude and frequency demodulation

In this paper, we view demodulation as an estimation problem in which a signal is fit with a sinusoid of time-varying amplitude and phase,

$$y_t = \Re \left( a_t \exp \left( i \phi_t \right) \right) + \epsilon_t. \tag{1}$$

The expression also includes a noise term which will be modeled as a zero-mean Gaussian with variance $\sigma_y^2$, that is $p(\epsilon_t) = \text{Norm}(\epsilon_t; 0, \sigma_y^2)$. We are interested in the situation where the IF of the sinusoid varies slowly around a mean value $\bar{\omega}$. In this case, the phase can be expressed in terms of the integrated mean frequency and a small perturbation, $\phi_t = \bar{\omega}t + \theta_t$.

Clearly, the problem of inferring $a_t$ and $\theta_t$ from $y_t$ is ill-posed, and results will depend on the specification of prior distributions over the amplitude and phase perturbation variables. Our goal in this paper is to specify such prior distributions directly, but this will require the development of new techniques to handle the resulting non-linearities. A simpler alternative is to generate the sinusoidal signal from a rotating two-dimensional phasor. For example, re-parametrizing the likelihood in terms of the components $x_{1,t} = a_t \cos(\theta_t)$ and $x_{2,t} = a_t \sin(\theta_t)$, yields a linear likelihood function

$$y_t = a_t \left( \cos(\bar{\omega}t) \cos(\theta_t) - \sin(\bar{\omega}t) \sin(\theta_t) \right) + \epsilon_t = \cos(\bar{\omega}t) x_{1,t} - \sin(\bar{\omega}t) x_{2,t} + \epsilon_t = \mathbf{w}_t^\mathsf{T} \mathbf{x}_t + \epsilon_t.$$

Here the phasor components, which have been collected into a vector $\mathbf{x}_t^\mathsf{T} = [x_{1,t}, x_{2,t}]$, are multiplied by time-varying weights, $\mathbf{w}_t^\mathsf{T} = [\cos(\bar{\omega}t), -\sin(\bar{\omega}t)]$. To complete the model, prior distributions can be now be specified over $\mathbf{x}_t$. One choice that results in a particularly simple inference algorithm is a Gaussian one-step auto-regressive (AR(1)) prior,

$$p(x_{k,t}|x_{k,t-1}) = \text{Norm}(x_{k,t}; \lambda x_{k,t-1}, \sigma_x^2). \tag{2}$$

When the dynamical parameter tends to unity ($\lambda \to 1$) and the dynamical noise variance to zero ($\sigma_x^2 \to 0$), the dynamics become very slow, and this slowness is inherited by the phase perturbations and amplitudes. This model is an instance of the Bayesian Spectrum Estimation (BSE) model [13] (when $\lambda = 1$), but re-interpreted in terms of amplitude- and frequency-modulated sinusoids, rather than fixed frequency basis functions. As the model is a linear Gaussian state space model, exact inference proceeds via the Kalman smoothing algorithm.

Before discussing the properties of BSE in the context of fitting amplitude- and frequency-modulated sinusoids, we derive an equivalent model by returning to the likelihood function (eq. 1). Now the full complex representation of the sinusoid is retained. As before, the real part corresponds to the observed data, but the imaginary part is now treated explicitly as missing data,

$$y_t = \Re \left( x_{1,t} \cos(\bar{\omega}t) - x_{2,t} \sin(\bar{\omega}t) + i x_{1,t} \sin(\bar{\omega}t) + i x_{2,t} \cos(\bar{\omega}t) \right) + \epsilon_t. \tag{3}$$

The new form of the likelihood function can be expressed in vector form, $y_t = [1, 0]\mathbf{z}_t + \epsilon_t$, using a new set of variables, $\mathbf{z}_t$, which are rotated versions of the original variables, $\mathbf{z}_t = \text{R}(\bar{\omega}t)\mathbf{x}_t$ where

$$\text{R}(\theta) = \begin{bmatrix} \cos(\theta) & -\sin(\theta) \\ \sin(\theta) & \cos(\theta) \end{bmatrix}. \tag{4}$$

An auto-regressive expression for the new variables, $\mathbf{z}_t$, can now be found using the fact that rotation matrices commute, $\text{R}(\theta_1 + \theta_2) = \text{R}(\theta_1)\text{R}(\theta_2) = \text{R}(\theta_2)\text{R}(\theta_1)$, together with expression for the dynamics of the original variables, $\mathbf{x}_t$ (eq. 2),

$$\mathbf{z}_t = \lambda \text{R}(\bar{\omega})\text{R}(\bar{\omega}(t-1))\mathbf{x}_{t-1} + \text{R}(\bar{\omega}t)\boldsymbol{\epsilon}_t = \lambda \text{R}(\bar{\omega})\mathbf{z}_{t-1} + \boldsymbol{\epsilon}_t' \tag{5}$$

where the noise is a zero mean Gaussian with covariance $\langle \boldsymbol{\epsilon}_t' \boldsymbol{\epsilon}_t'^\mathsf{T} \rangle = \text{R}(\bar{\omega}t)\langle \boldsymbol{\epsilon}_t \boldsymbol{\epsilon}_t^\mathsf{T} \rangle \text{R}^\mathsf{T}(\bar{\omega}t) = \sigma_x^2 \text{I}$. This equivalent formulation of the BSE model is called the Probabilistic Phase Vocoder (PPV) [14]. Again exact inference is possible using the Kalman smoothing algorithm.

## 1.2 Problems with simple models for probabilistic amplitude and frequency demodulation

BSE-PPV is used to demodulate synthetic and natural signals in Figs. 1, 2 and 7. The decomposition is compared to the Hilbert method. These examples immediately reveal several problems with BSE-PPV. Perhaps most unsatisfactory is the fact that the IF estimates are often ill behaved, to the extent that they go negative, especially in regions where the amplitude of the signal is low. It is easy to understand why this occurs by considering the prior distribution over amplitude and phase implied by our choice of prior distribution over $\mathbf{x}_t$ (or equivalently over $\mathbf{z}_t$),

$$p(a_t, \phi_t | a_{t-1}, \phi_{t-1}) = \frac{a_t}{2\pi\sigma_x^2} \exp\left[-\frac{1}{2\sigma_x^2}\left(a_t^2 + \lambda^2 a_{t-1}^2\right) + \frac{\lambda}{\sigma_x^2} a_t a_{t-1} \cos(\phi_t - \phi_{t-1} - \bar{\omega})\right]. \quad (6)$$

Phase and amplitude are dependent in the implied distribution, which is conditionally a uniform distribution over phase when the amplitude is zero and a strongly peaked von Mises distribution [15] when the amplitude is large. Consequently, the model favors more highly variable IFs at low amplitudes. In some applications this may be desirable, but for signals like sounds it presents a problem. First it may assign substantial probability to unphysical negative IFs. Second, the same noiseless signal at different intensities will yield different estimated IF content. Third, the complex coupling makes it difficult to select domain-appropriate time-scale parameters. Consideration of IF reveals yet another problem. When the phase-perturbations vary slowly ($\lambda \to 1$), there is no correlation between successive IFs ($\langle\omega_t\omega_{t-1}\rangle - \langle\omega_t\rangle\langle\omega_{t-1}\rangle \to 0$). One of the main goals of the model was to capture correlated IFs through time, and the solution is to move to priors with higher order temporal dependencies.

In the next section we will propose a new model for PAFD which addresses these problems, retaining the same likelihood function, but modifying the prior to include independent distributions over the phase and amplitude variables.

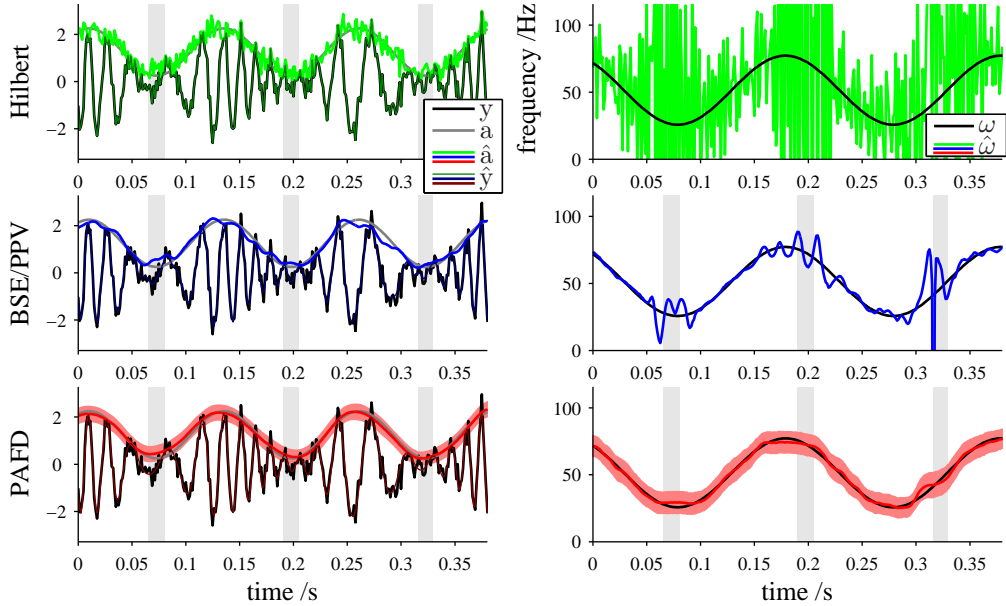

Figure 1: Comparison of AFD methods on a sinusoidally amplitude- and frequency-modulated sinusoid in broad-band noise. Estimated values are shown in red. The gray areas show the region where the true amplitude falls below the noise floor ($a < \sigma_y$) and the estimates become less accurate. See section 4 for details.

## 2 PAFD using Auto-regressive and generalized von Mises distributions

We have argued that the amplitude and phase variables in a model for PAFD should be independently parametrized, but that this introduces difficulties as the likelihood is highly non-linear in these variables. This section and the next develop the tools necessary to handle this non-linearity.

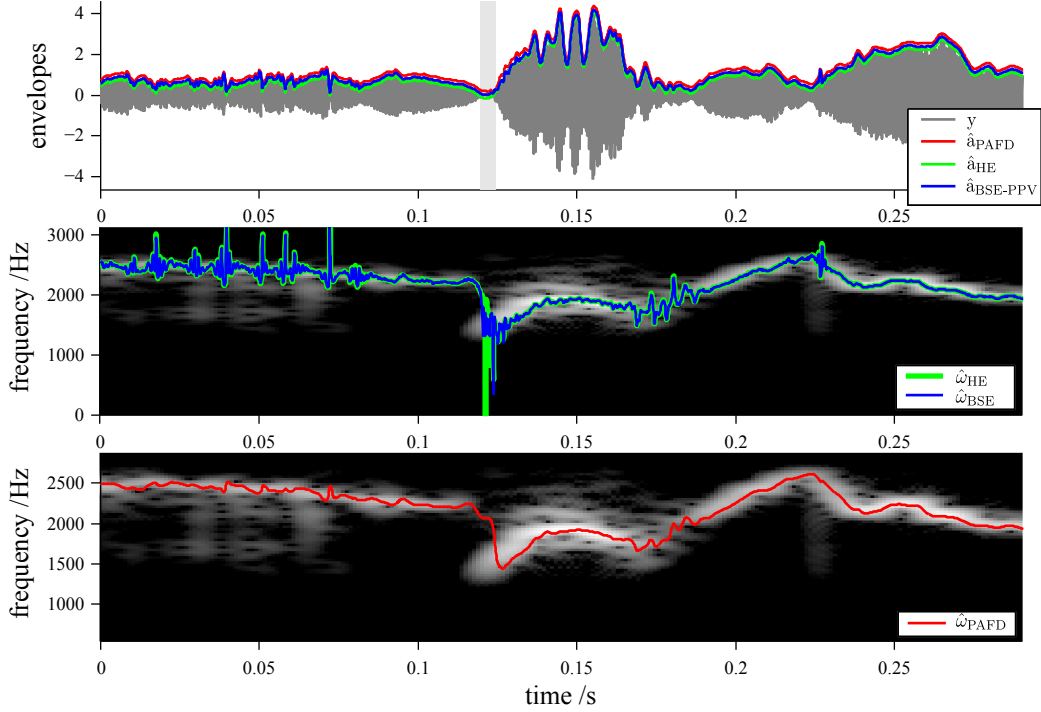

Figure 2: AFD of a starling song. Top: The original waveform with estimated envelopes, shifted apart vertically to aid visualization. The light gray bar indicates the problematic low amplitude region. Bottom panels: IF estimates superposed onto the spectrum of the signal. PAFD tracks the FM/AM well, but the other methods have artifacts.

An important initial consideration is whether to use a representation for phase which is wrapped, $\theta \in (-\pi, \pi]$, or unwrapped, $\theta \in \mathbb{R}$. Although the latter has the advantage of implying simpler dynamics, it leads to a potential infinity of local modes at multiples of $2\pi$ making inference extremely difficult. It is therefore necessary to work with wrapped phases and a sensible starting point for a prior is thus the von Mises distribution,

$$p(\theta|\mathrm{k}, \mu) = \frac{1}{2\pi I_0(\mathrm{k})} \exp(\mathrm{k}\cos(\theta - \mu)) = \mathrm{vonMises}(\theta; \mathrm{k}, \mu). \tag{7}$$

The two parameters, the concentration (k) and the mean ($\mu$), determine the circular variance and mean of the distribution respectively. The normalizing constant is given by a modified Bessel function of the second kind, $I_0(\mathrm{k})$. Crucially for our purposes, the von Mises distribution can be obtained by taking a bivariate isotropic Gaussian with an arbitrary mean, and conditioning onto the unit-circle (this connects with BSE-PPV, see eq. 6). The Generalized von Mises distribution is formed in an identical way when the bivariate Gaussian is anisotropic [16]. These constructions suggest a simple extension to time-series data by conditioning a temporal bivariate Gaussian time-series onto the unit circle at all sample times. For example, when two independent Gaussian AR(2) distributions are used to construct the prior we have,

$$p(\mathrm{x}_{1:2,1:T}) \propto \prod_{t=1}^{T} \mathbf{1}(\mathrm{x}_{1,t}^2 + \mathrm{x}_{2,t}^2 = 1) \prod_{m=1}^{2} \mathrm{Norm}(\mathrm{x}_{m,t}; \lambda_1 \mathrm{x}_{m,t-1} + \lambda_2 \mathrm{x}_{m,t-2}, \sigma_\mathrm{x}^2). \tag{8}$$

where $\mathbf{1}(\mathrm{x}_{1,t}^2 + \mathrm{x}_{2,t}^2 = 1)$ is an indicator function representing the unit circle constraint. Upon a change of variables $\mathrm{x}_{1,t} = \cos(\theta_t)$, $\mathrm{x}_{2,t} = \sin(\theta_t)$ this yields,

$$p(\theta_{1:T}|\mathrm{k}_1, \mathrm{k}_2) \propto \prod_{t=1}^{T} \exp\left(\mathrm{k}_1 \cos(\theta_t - \theta_{t-1}) + \mathrm{k}_2 \cos(\theta_t - \theta_{t-2})\right), \tag{9}$$

where $k_1 = \lambda_1(1 - \lambda_2)/\sigma_x^2$ and $k_2 = \lambda_2/\sigma_x^2$. One of the attractive features of this prior is that when it is combined with the likelihood (eq. 1) the resulting posterior distribution over phase variables is a temporal version of the Generalized von Mises distribution. That is, it can be expressed as a bivariate anisotropic Gaussian, which is constrained to the unit circle. It is this representation which will prove essential for inference.

Having established a candidate prior over phases, we turn to the amplitude variables. With one eye upon the fact that the prior over phases can be interpreted as product of a Gaussian and a constraint, we employ a prior of a similar form for the amplitude variables; a truncated Gaussian AR($\tau$) process,

$$p(a_{1:T}|\lambda_{1:\tau}, \sigma^2) \propto \prod_{t=1}^{T} \mathbf{1}(a_t \geq 0) \operatorname{Norm}\left(a_t; \sum_{t'=1}^{\tau} \lambda_{t'} a_{t-t'}, \sigma^2\right). \tag{10}$$

The model formed from equations 1, 9 and 10 will be termed Probabilistic Amplitude and Frequency Demodulation. PAFD is closely related to the BSE-PPV model [13, 14]. Moreover, when the phase variables are drawn from a uniform distribution ($k_1 = k_2 = 0$) it reduces to the convex amplitude demodulation model [17], which itself is a form of probabilistic amplitude demodulation [18, 19, 20]. The AR prior over phases has also been used in a regression setting [21].

## 3 Inference via expectation propagation

The PAFD model introduced in the last section contains three separate types of non-linearity: the multiplicative interaction in the likelihood, the unit circle constraint, and the positivity constraint. Of these, it is the circular constraint which is most challenging as the development of general purpose machine learning methods for handling hard, non-convex constraints is an open research problem. Following [22], we propose a novel method which uses expectation propagation (EP) [23] to replace the hard constraints with soft, local, Gaussian approximations which are iteratively refined.

In order to apply EP, the model is first rewritten into a simpler form. Making use of the fact that an AR($\tau$) process can be rewritten as an equivalent multi-dimensional AR(1) model with $\tau$ states, we concatenate the latent variables into an augmented state vector, $\mathbf{s}_t^\mathsf{T} = [a_t, a_{t-1}, \ldots, a_{t-\tau+1}, x_{1,t}, x_{2,t}, x_{1,t-1}, x_{2,t-1}]$, and express the model as a product of clique potentials in terms of this variable,

$$p(y_{1:T}, \mathbf{s}_{1:T}) \propto \prod_{t=1}^{T} \pi_t(\mathbf{s}_t, \mathbf{s}_{t-1}) \psi_t(s_{1,t}, s_{1+\tau,t}, s_{2+\tau,t}), \text{ where } \pi_t(\mathbf{s}_t, \mathbf{s}_{t-1}) = \operatorname{Norm}(\mathbf{s}_t; \Lambda_s \mathbf{s}_{t-1}, \Sigma_s),$$

$$\psi_t(a_t, x_{1,t}, x_{2,t}) = \operatorname{Norm}\left(y_t; a_t(\cos(\bar{\omega}t)x_{1,t} - \sin(\bar{\omega}t)x_{2,t}), \sigma_y^2\right) \mathbf{1}(a_t \geq 0)\mathbf{1}(x_{1,t}^2 + x_{2,t}^2 = 1).$$

(See the supplementary material for details of the dynamical matrices $\Lambda_s$ and $\Sigma_s$). In this new form the constraints have been incorporated with the non-linear likelihood into the potential $\psi_t$, leaving a standard Gaussian dynamical potential $\pi_t(\mathbf{s}_t, \mathbf{s}_{t-1})$. Using EP we approximate the posterior distribution using a product of forward, backward and constrained-likelihood messages [24],

$$q(\mathbf{s}_{1:T}) = \prod_{t=1}^{T} \alpha_t(\mathbf{s}_t)\beta_t(\mathbf{s}_t)\tilde{\psi}_t(a_{1,t}, x_{1,t}, x_{2,t}) = \prod_{t=1}^{T} q_t(\mathbf{s}_t). \tag{11}$$

The messages should be interpreted as follows: $\alpha_t(\mathbf{s}_t)$ is the effect of $\pi_t(\mathbf{s}_{t-1}, \mathbf{s}_t)$ and $q(\mathbf{s}_{t-1})$ on the belief $q(\mathbf{s}_t)$, whilst $\beta_t(\mathbf{s}_t)$ is the effect of $\pi_{t+1}(\mathbf{s}_t, \mathbf{s}_{t+1})$ and $q(\mathbf{s}_{t+1})$ on the belief $q(\mathbf{s}_t)$. Finally, $\tilde{\psi}_t(a_{1,t}, x_{1,t}, x_{2,t})$ is the effect of the likelihood and the constraints on the belief $q(\mathbf{s}_t)$. All of these messages will be un-normalized Gaussians. The updates for the messages can be found by removing the messages from $q(\mathbf{s}_{1:T})$ that correspond to the effect of a particular potential. These messages are replaced by the corresponding potential. The deleted messages are then updated by moment matching the two distributions. The updates for the forward and backward messages are a straightforward application of EP and result in updates that are nearly identical to those used for Kalman smoothing. The updates for the constrained likelihood potential are more complicated:

$$\text{update } \tilde{\psi}_t \text{ such that } q(\mathbf{x}_t) \overset{\text{MOM}}{=} \hat{p}_\psi(\mathbf{s}_t) = \alpha_t(\mathbf{s}_t)\beta_t(\mathbf{s}_t)\psi_t(a_t, x_{1,t}, x_{2,t}). \tag{12}$$

The difficulty is the moment computation which we evaluate in two stages. First, we integrate over the amplitude variable, which involves computing the moments of a truncated Gaussian and

is therefore computationally efficient. Second, we numerically integrate over the one dimensional phase variable. For the details we again refer the reader to the supplementary material.

A standard forward-backward message update schedule was used. Adaptive damping improved the numerical stability of the algorithm substantially. The computational complexity of PAFD is $\mathrm{O}\big(T(N + \tau^3)\big)$ where $N$ are the number of points used to compute the integral over the phase variable. For the experiments we used a second order process over the amplitude variables ($\tau = 2$) and $N = 1000$ integration points. In this case, the 16-32 forward-backward passes required for convergence took one minute on a modern laptop computer for signals of length $T = 1000$.

## 4  Application to synthetic signals

One of the main challenges posed by the evaluation of AFD algorithms is that the ground truth for real-world signals is unknown. This means that a quantitative comparison of different schemes must take an indirect approach. The first set of evaluations presented here uses synthetic signals, for which the ground truth is known. In particular, we consider amplitude- and frequency-modulated sinusoids, $y_t = a_t \cos(\theta_t)$ where $a_t = 1 + \sin(2\pi f_a t)$ and $\frac{1}{2\pi}\frac{d\theta}{dt} = \bar{f} + \Delta_f \sin(2\pi f_f t)$, which have been corrupted by Gaussian noise. Fig. 1 compares AFD of one such signal ($\bar{f} = 50$Hz, $f_a = 8$Hz, $f_f = 5$Hz and $\Delta_f = 25$Hz) by the Hilbert, BSE-PPV and PAFD methods. Fig. 3 summarizes the results at different noise levels in terms of the signal to noise ratio (SNR) of the estimated variables and the reconstructed signal, i.e. $\mathrm{SNR}(a) = 10\log_{10}\sum_{t=1}^{T} a_t^2 - 10\log_{10}\sum_{t=1}^{T}\left(a_t - \hat{a}_t\right)^2$. PAFD consistently outperforms the other methods by this measure. Furthermore, Fig. 4 demonstrates that PAFD can be used to accurately reconstruct missing sections of this signal, outperforming BSE-PPV.

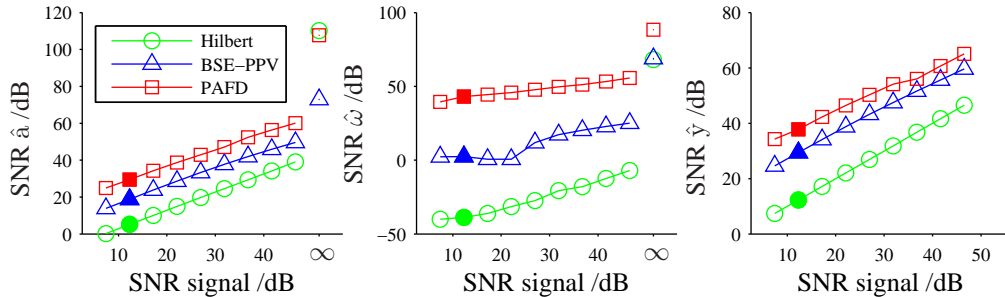

Figure 3: Noisy synthetic data. SNR of estimated variables as a function of the SNR of the signal. Envelopes (left), IFs (center) and denoised signal (right). Solid markers denote examples in Fig. 1.

## 5  Application to real world signals

Having validated PAFD on simple synthetic examples, we now consider real-world signals. Birdsong is used as a prototypical signal as it has strong frequency-modulation content. We isolate a 300ms component of a starling song using a bandpass filter and apply AFD. Fig. 2 shows that PAFD can track the underlying frequency modulation even though there is noise in the signal which causes the other methods to fail. This example forms the basis of two important robustness and consistency tests. In the first, spectrally matched noise is added to the signal and the IFs and amplitudes are re-estimated and compared to those derived from the clean signal. Fig. 5 shows that the PAFD method is considerably more robust to this manipulation than both the Hilbert and BSE-PPV methods. In the second test, regions of the signal are removed and the model's predictions for the missing regions are compared to the estimates derived from the clean signal (see fig. 6). Once again PAFD is more accurate. As a final test of PAFD we consider the important neuroscientific task of estimating the phase, equivalently the IF, of theta oscillations in an EEG signal. The EEG signal typically contains broadband noise and so a conventional analysis applies a band-pass filter before using the Hilbert method to estimate the IF. Although this improves the estimates markedly, the noise component cannot be completely eradicated which leads to artifacts in the IF estimates (see Fig. 7). In contrast

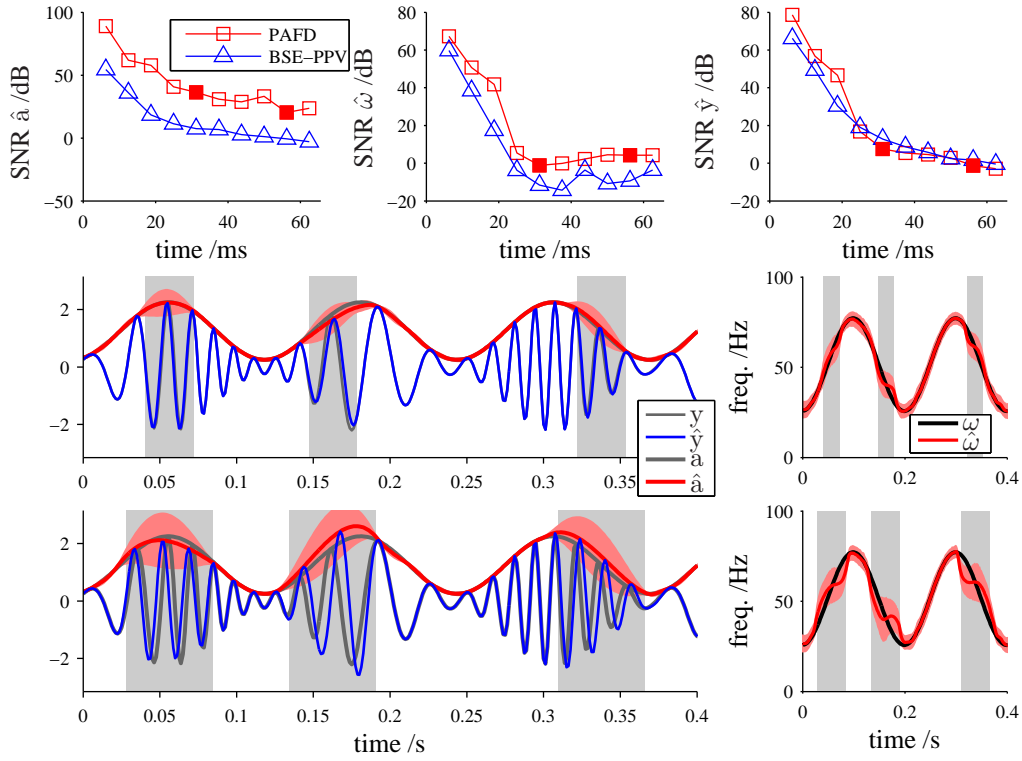

Figure 4: Missing synthetic data experiments. TOP: SNR of estimated variables as a function of gap duration in the input signal. Envelopes (left), IFs (center) and denoised signal (right). Solid markers indicate the examples shown in the bottom rows of the figure. BOTTOM: Two examples of PAFD reconstruction. Light gray regions indicate missing sections of the signal.

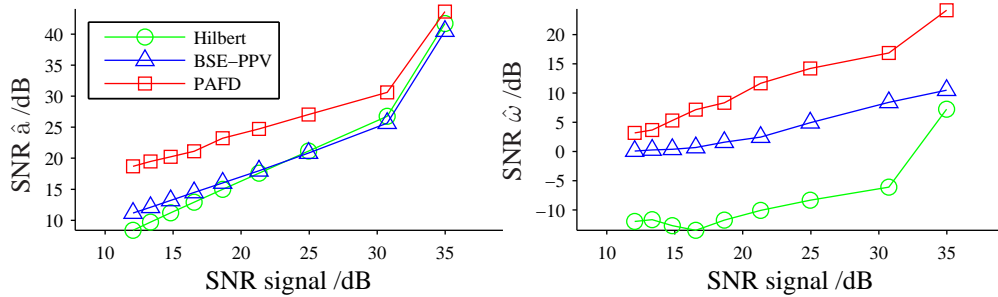

Figure 5: Noisy bird song experiments. SNR of estimated variables as compared to those estimated from the clean signal, as a function of the SNR of the input signal. Envelopes (left), IFs (right).

PAFD returns sensible estimates from both the filtered and original signal. Critically, both estimates are similar to one another suggesting the new estimation scheme is reliable.

# 6 Conclusion

Amplitude and frequency demodulation is a difficult, ill-posed estimation problem. We have developed a new inferential solution called probabilistic amplitude and frequency demodulation which employs a von Mises time-series prior over phase, constructed by conditioning a bivariate Gaussian auto-regressive distribution onto the unit circle. The construction naturally leads to an expectation propagation inference scheme which approximates the hard constraints using soft local Gaussians.

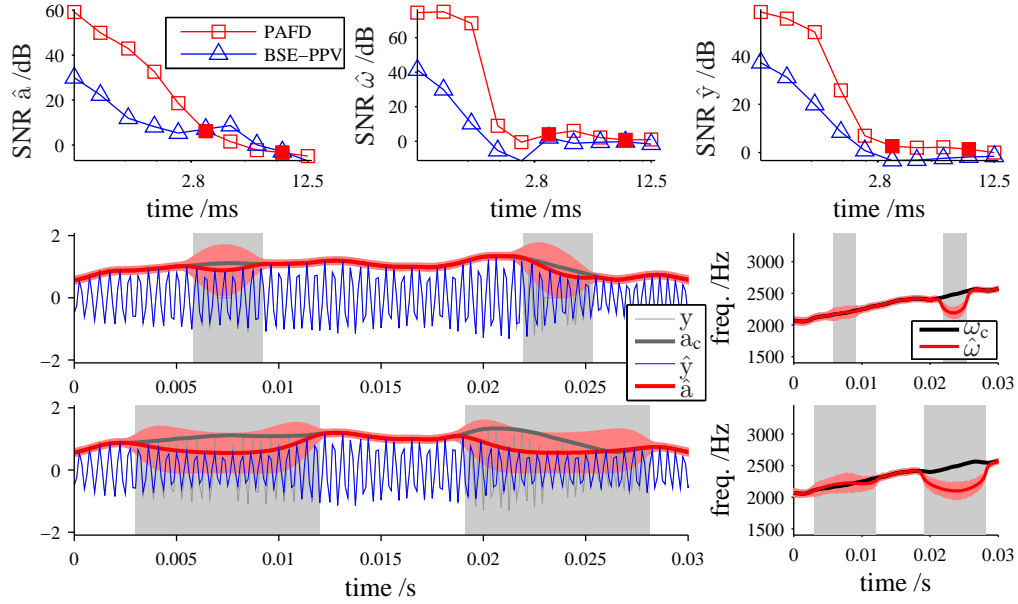

Figure 6: Missing natural data experiments. TOP: SNR of estimated variables as a function of gap duration in the input signal. Envelopes (left), IFs (center) and denoised signal (right). Solid markers indicate the examples shown in the bottom rows of the figure. BOTTOM: Two examples of PAFD reconstruction. Light gray regions indicate missing sections of the signal.

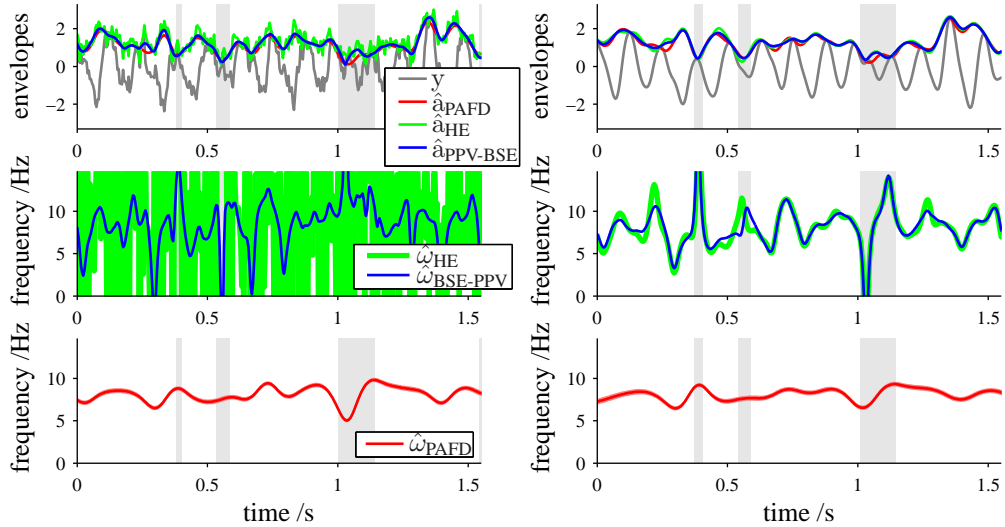

Figure 7: Comparison of AFD methods on EEG data. The left hand side shows estimates derived from the raw EEG signal, whilst the right shows estimates derived from a band-pass filtered version. The gray areas show the region where the true amplitude falls below the noise floor ($a < \sigma_y$), where conventional methods fail.

We have demonstrated the utility of the new method on synthetic and natural signals, where it outperformed conventional approaches. Future research will consider extensions of the model to multiple sinusoids, and learning the model parameters so that the algorithm can adapt to novel signals.

### Acknowledgments

Richard Turner was funded by the EPRC, and Maneesh Sahani by the Gatsby Charitable Foundation.

# References

[1] P. J. Loughlin and B. Tacer. On the amplitude- and frequency-modulation decomposition of signals. *The Journal of the Acoustical Society of America*, 100(3):1594–1601, 1996.

[2] J. L. Flanagan. Parametric coding of speech spectra. *The Journal of the Acoustical Society of America*, 68:412–419, 1980.

[3] P. Clark and L.E. Atlas. Time-frequency coherent modulation filtering of nonstationary signals. *Signal Processing, IEEE Transactions on*, 57(11):4323 –4332, nov. 2009.

[4] J. L. Flanagan and R. M. Golden. Phase vocoder. *Bell System Technical Journal*, pages 1493–1509, 1966.

[5] S. M. Schimmel. *Theory of Modulation Frequency Analysis and Modulation Filtering, with Applications to Hearing Devices*. PhD thesis, University of Washington, 2007.

[6] L. E. Atlas and C. Janssen. Coherent modulation spectral filtering for single-channel music source separation. In *Proceedings of the IEEE Conference on Acoustics Speech and Signal Processing*, 2005.

[7] Z. M. Smith, B. Delgutte, and A. J. Oxenham. Chimaeric sounds reveal dichotomies in auditory perception. *Nature*, 416(6876):87–90, 2002.

[8] J. Dugundji. Envelopes and pre-envelopes of real waveforms. *IEEE Transactions on Information Theory*, 4:53–57, 1958.

[9] O. Ghitza. On the upper cutoff frequency of the auditory critical-band envelope detectors in the context of speech perception. *The Journal of the Acoustical Society of America*, 110(3):1628–1640, 2001.

[10] F. G. Zeng, K. Nie, S. Liu, G. Stickney, E. Del Rio, Y. Y. Kong, and H. Chen. On the dichotomy in auditory perception between temporal envelope and fine structure cues (L). *The Journal of the Acoustical Society of America*, 116(3):1351–1354, 2004.

[11] D. Vakman. On the analytic signal, the Teager-Kaiser energy algorithm, and other methods for defining amplitude and frequency. *IEEE Journal of Signal Processing*, 44(4):791–797, 1996.

[12] G. Girolami and D. Vakman. Instantaneous frequency estimation and measurement: a quasi-local method. *Measurement Science and Technology*, 13(6):909–917, 2002.

[13] Y. Qi, T. P. Minka, and R. W. Picard. Bayesian spectrum estimation of unevenly sampled nonstationary data. In *International Conference on Acoustics, Speech, and Signal Processing*, 2002.

[14] A. T. Cemgil and S. J. Godsill. Probabilistic Phase Vocoder and its application to Interpolation of Missing Values in Audio Signals. In *13th European Signal Processing Conference*, Antalya/Turkey, 2005.

[15] C. Bishop. *Pattern Recognition and Machine Learning*. Springer, 2006.

[16] R. Gatto and S. R. Jammalamadaka. The generalized von mises distribution. *Statistical Methodology*, 4:341–353, 2007.

[17] G. Sell and M. Slaney. Solving demodulation as an optimization problem. *IEEE Transactions on Audio, Speech and Language Processing*, 18:2051–2066, November 2010.

[18] R. E. Turner and M. Sahani. Probabilistic amplitude demodulation. In *Independent Component Analysis and Signal Separation*, pages 544–551, 2007.

[19] R. E. Turner and M. Sahani. Statistical inference for single- and multi-band probabilistic amplitude demodulation. In *Proceedings of the IEEE International Conference on Acoustics, Speech, and Signal Processing (ICASSP)*, pages 5466–5469, 2010.

[20] R. E. Turner and M. Sahani. Demodulation as probabilistic inference. *IEEE Transactions on Audio, Speech and Language Processing*, 2011.

[21] J. Breckling. *The analysis of directional time series: Application to wind speed and direction*. Springer-Verlag, 1989.

[22] J. P. Cunningham. *Algorithms for Understanding Motor Cortical Processing and Neural Prosthetic Systems*. PhD thesis, Stanford University, Department of Electrical Engineering, (Stanford, California, USA, 2009.

[23] T. Minka. *A family of algorithms for approximate Bayesian inference*. PhD thesis, MIT Media Lab, 2001.

[24] T. Heskes and O. Zoeter. Expectation propagation for approximate inference in dynamic bayesian networks. In *A. Darwiche and N. Friedman*, pages 216–233. Morgan Kaufmann Publishers, 2002.

